# ALVINN:
# AN AUTONOMOUS LAND VEHICLE IN A
# NEURAL NETWORK

Dean A. Pomerleau
Computer Science Department
Carnegie Mellon University
Pittsburgh, PA 15213

## ABSTRACT

ALVINN (Autonomous Land Vehicle In a Neural Network) is a 3-layer back-propagation network designed for the task of road following. Currently ALVINN takes images from a camera and a laser range finder as input and produces as output the direction the vehicle should travel in order to follow the road. Training has been conducted using simulated road images. Successful tests on the Carnegie Mellon autonomous navigation test vehicle indicate that the network can effectively follow real roads under certain field conditions. The representation developed to perform the task differs dramatically when the network is trained under various conditions, suggesting the possibility of a novel adaptive autonomous navigation system capable of tailoring its processing to the conditions at hand.

## INTRODUCTION

Autonomous navigation has been a difficult problem for traditional vision and robotic techniques, primarily because of the noise and variability associated with real world scenes. Autonomous navigation systems based on traditional image processing and pattern recognition techniques often perform well under certain conditions but have problems with others. Part of the difficulty stems from the fact that the processing performed by these systems remains fixed across various driving situations.

Artificial neural networks have displayed promising performance and flexibility in other domains characterized by high degrees of noise and variability, such as handwritten character recognition [Jackel et al., 1988] [Pawlicki et al., 1988] and speech recognition [Waibel et al., 1988]. ALVINN (Autonomous Land Vehicle In a Neural Network) is a connectionist approach to the navigational task of road following. Specifically, ALVINN is an artificial neural network designed to control the NAVLAB, the Carnegie Mellon autonomous navigation test vehicle.

## NETWORK ARCHITECTURE

ALVINN's current architecture consists of a single hidden layer back-propagation network

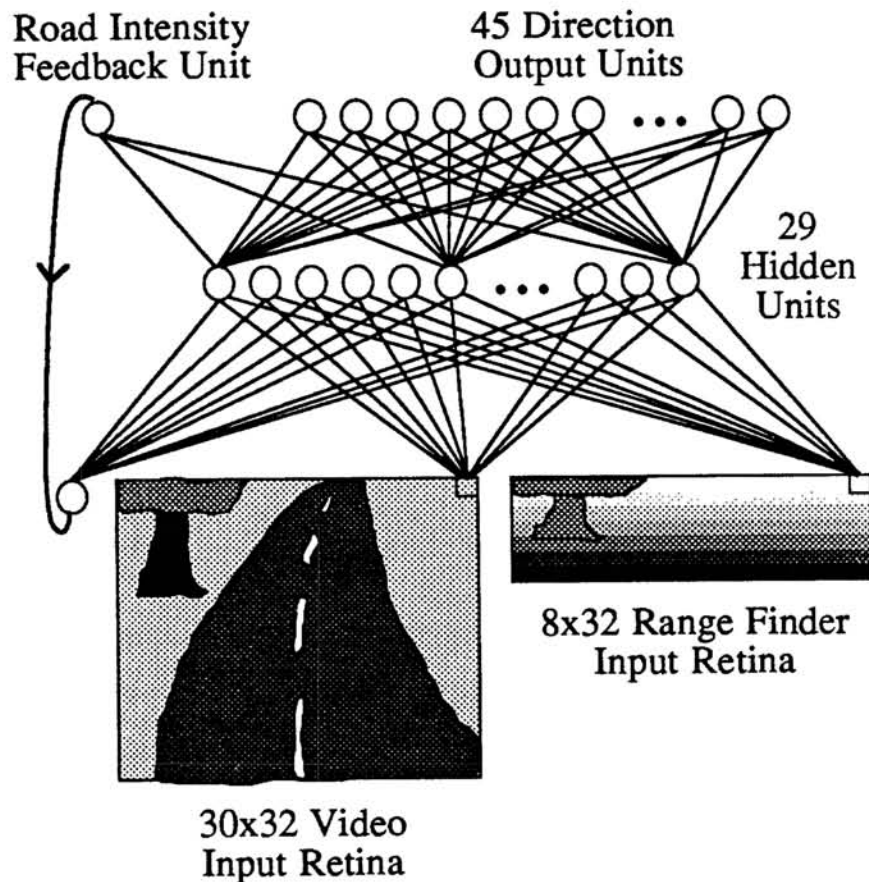

Road Intensity
Feedback Unit

45 Direction
Output Units

29
Hidden
Units

8x32 Range Finder
Input Retina

30x32 Video
Input Retina

Figure 1: ALVINN Architecture

(See Figure 1). The input layer is divided into three sets of units: two "retinas" and a single intensity feedback unit. The two retinas correspond to the two forms of sensory input available on the NAVLAB vehicle; video and range information. The first retina, consisting of 30x32 units, receives video camera input from a road scene. The activation level of each unit in this retina is proportional to the intensity in the blue color band of the corresponding patch of the image. The blue band of the color image is used because it provides the highest contrast between the road and the non-road. The second retina, consisting of 8x32 units, receives input from a laser range finder. The activation level of each unit in this retina is proportional to the proximity of the corresponding area in the image. The road intensity feedback unit indicates whether the road is lighter or darker than the non-road in the previous image. Each of these 1217 input units is fully connected to the hidden layer of 29 units, which is in turn fully connected to the output layer.

The output layer consists of 46 units, divided into two groups. The first set of 45 units is a linear representation of the turn curvature along which the vehicle should travel in order to head towards the road center. The middle unit represents the "travel straight ahead" condition while units to the left and right of the center represent successively sharper left and right turns. The network is trained with a desired output vector of all zeros except for a "hill" of activation centered on the unit representing the correct turn curvature, which is the curvature which would bring the vehicle to the road center 7 meters ahead of its current position. More specifically, the desired activation levels for

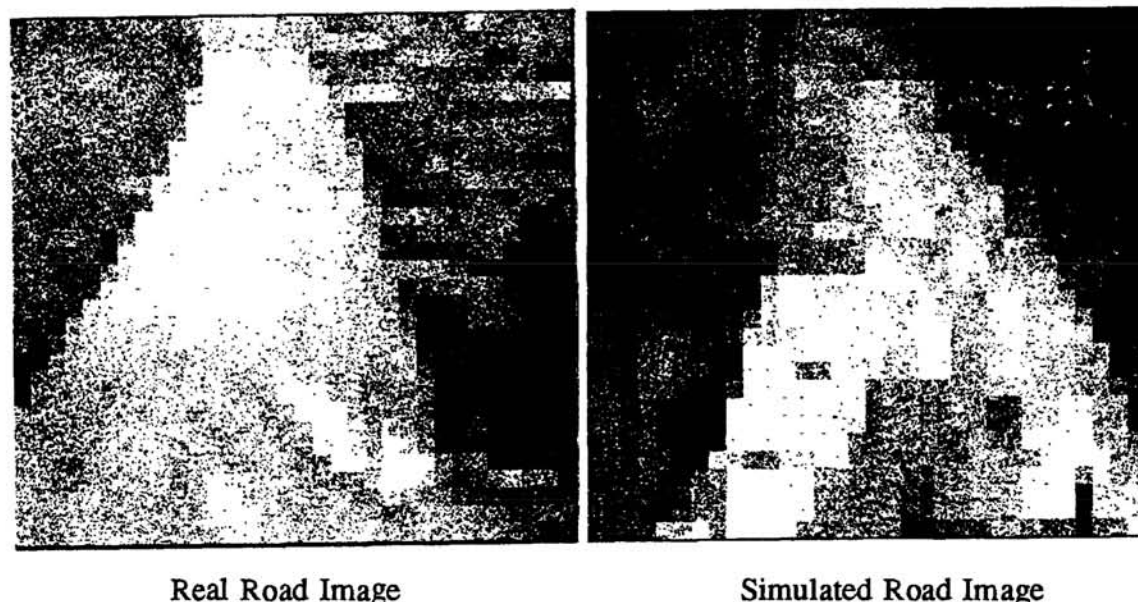

Real Road Image                     Simulated Road Image

Figure 2: Real and simulated road images

the nine units centered around the correct turn curvature unit are 0.10, 0.32, 0.61, 0.89, 1.00, 0.89, 0.61, 0.32 and 0.10. During testing, the turn curvature dictated by the network is taken to be the curvature represented by the output unit with the highest activation level.

The final output unit is a road intensity feedback unit which indicates whether the road is lighter or darker than the non-road in the current image. During testing, the activation of the output road intensity feedback unit is recirculated to the input layer in the style of Jordan [Jordan, 1988] to aid the network's processing by providing rudimentary information concerning the relative intensities of the road and the non-road in the previous image.

## TRAINING AND PERFORMANCE

Training on actual road images is logistically difficult, because in order to develop a general representation, the network must be presented with a large number of training exemplars depicting roads under a wide variety of conditions. Collection of such a data set would be difficult, and changes in parameters such as camera orientation would require collecting an entirely new set of road images. To avoid these difficulties we have developed a simulated road generator which creates road images to be used as training exemplars for the network. Figure 2 depicts the video images of one real and one artificial road. Although not shown in Figure 2, the road generator also creates corresponding simulated range finder images. At the relatively low resolution being used it is difficult to distinguish between real and simulated roads.

Network training is performed using these artificial road "snapshots" and the Warp back-

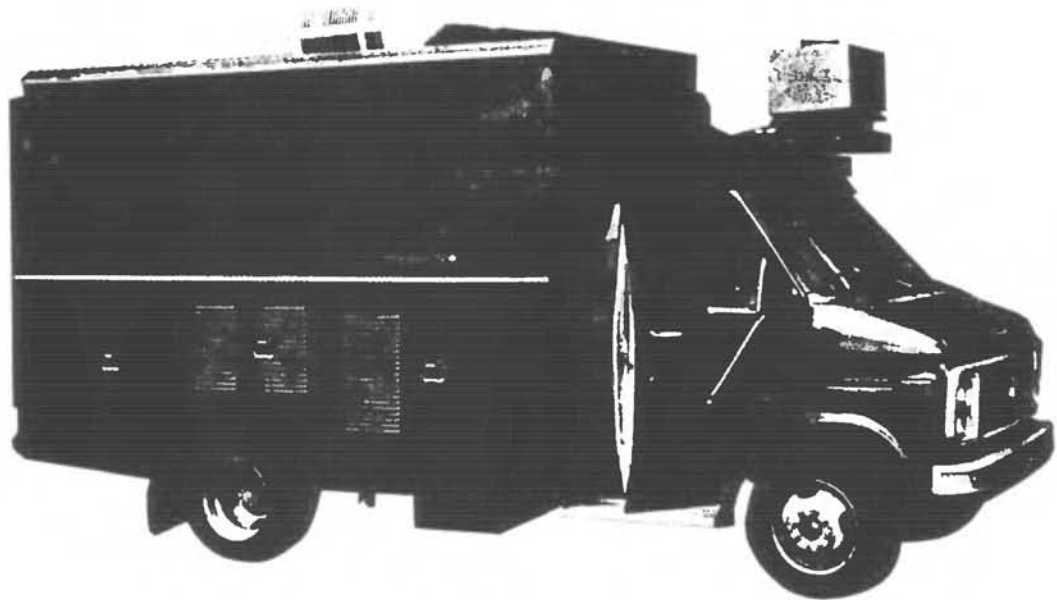

Figure 3: NAVLAB, the CMU autonomous navigation test vehicle.

propagation simulator described in [Pomerleau et al., 1988]. Training involves first creating a set of 1200 road snapshots depicting roads with a wide variety of retinal orientations and positions, under a variety of lighting conditions and with realistic noise levels. Back-propagation is then conducted using this set of exemplars until only asymptotic performance improvements appear likely. During the early stages of training, the input road intensity unit is given a random activation level. This is done to prevent the network from merely learning to copy the activation level of the input road intensity unit to the output road intensity unit, since their activation levels should almost always be identical because the relative intensity of the road and the non-road does not often change between two successive images. Once the network has developed a representation that uses image characteristics to determine the activation level for the output road intensity unit, the network is given as input whether the road would have been darker or lighter than the non-road in the previous image. Using this extra information concerning the relative brightness of the road and the non-road, the network is better able to determine the correct direction for the vehicle to travel.

After 40 epochs of training on the 1200 simulated road snapshots, the network correctly dictates a turn curvature within two units of the correct answer approximately 90% of the time on novel simulated road images. The primary testing of the ALVINN's performance has been conducted on the NAVLAB (See Figure 3). The NAVLAB is a modified Chevy van equipped with 3 Sun computers, a Warp, a video camera, and a laser range finder, which serves as a testbed for the CMU autonomous land vehicle project [Thorpe et al., 1987]. Performance of the network to date is comparable to that achieved by the best traditional vision-based autonomous navigation algorithm at CMU under the limited conditions tested. Specifically, the network can accurately drive the NAVLAB at a speed of 1/2 meter per second along a 400 meter path through a wooded

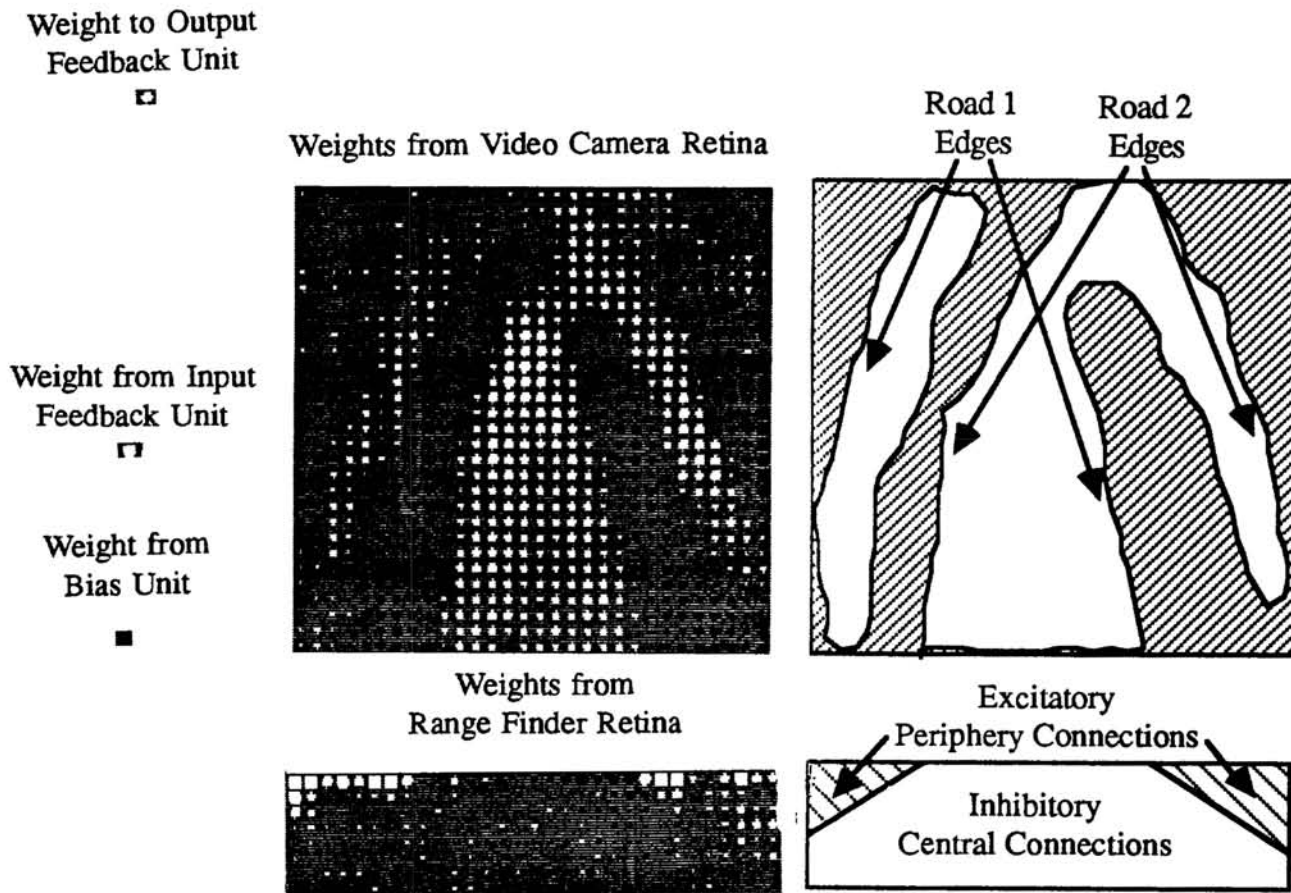

Weights to Direction Output Units

Weight to Output
Feedback Unit

Weights from Video Camera Retina

Road 1
Edges

Road 2
Edges

Weight from Input
Feedback Unit

Weight from
Bias Unit

Weights from
Range Finder Retina

Excitatory
Periphery Connections

Inhibitory
Central Connections

Figure 4: Diagram of weights projecting to and from a typical hidden unit in a network trained on roads with a fixed width. The schematics on the right are aids for interpretation.

area of the CMU campus under sunny fall conditions. Under similar conditions on the same course, the ALV group at CMU has recently achieved similar driving accuracy at a speed of one meter per second by implementing their image processing autonomous navigation algorithm on the Warp computer. In contrast, the ALVINN network is currently simulated using only an on-board Sun computer, and dramatic speedups are expected when tests are performed using the Warp.

## NETWORK REPRESENTATION

The representation developed by the network to perform the road following task depends dramatically on the characteristics of the training set. When trained on examples of roads with a fixed width, the network develops a representations consisting of overlapping road filters. Figure 4 is a diagram of the weights projecting to and from a single hidden unit in such a network.

As indicated by the weights to and from the feedback units, this hidden unit expects the road to be lighter than the non-road in the previous image and supports the road being lighter than the non-road in the current image. More specifically, the weights from the

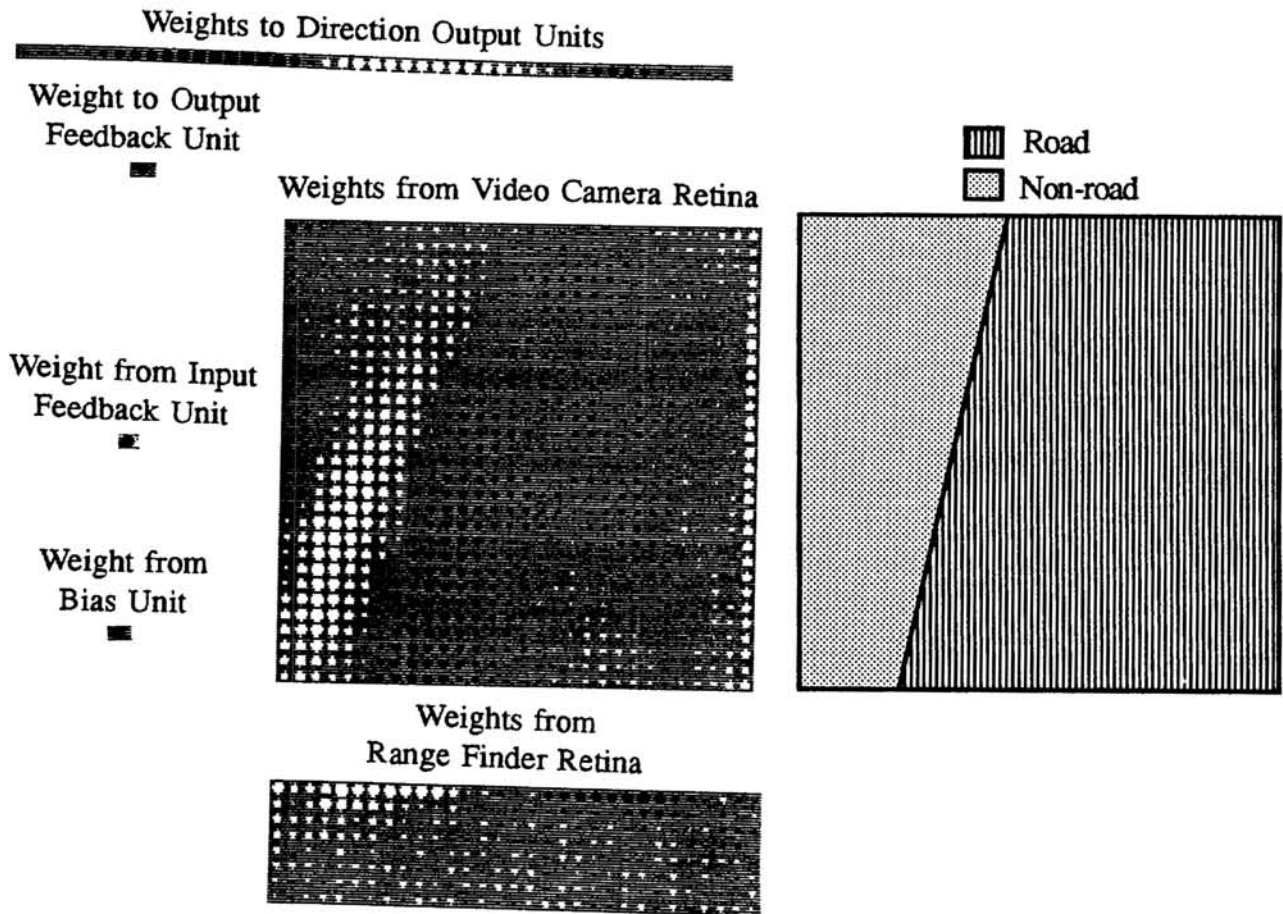

Figure 5: Diagram of weights projecting to and from a typical hidden unit in a network trained on roads with different widths.

video camera retina support the interpretation that this hidden unit is a filter for two light roads, one slightly left and the other slightly right or center (See schematic to the right of the weights from the video retina in Figure 4). This interpretation is also supported by the weights from the range finder retina, which has a much wider field of view than the video camera. This hidden unit is excited if there is high range activity (i.e. obstacles) in the periphery and inhibited if there is high range activity in the central region of the scene where this hidden unit expects the road to be (See schematic to the right of the weights from the range finder retina in Figure 4). Finally, the two road filter interpretation is reflected in the weights from this hidden unit to the direction output units. Specifically, this hidden unit has two groups of excitatory connections to the output units, one group dictating a slight left turn and the other group dictating a slight right turn. Hidden units which act as filters for 1 to 3 roads are the representation structures most commonly developed when the network is trained on roads with a fixed width.

The network develops a very different representation when trained on snapshots with widely varying road widths. A typical hidden unit from this type of representation is depicted in figure 5. One important feature to notice from the feedback weights is that this unit is filtering for a road which is darker than the non-road. More importantly, it is evident from the video camera retina weights that this hidden unit is a filter solely for the left edge of the road (See schematic to the right of the weights from the range finder

retina in Figure 5). This hidden unit supports a rather wide range of travel directions. This is to be expected, since the correct travel direction for a road with an edge at a particular location varies substantially depending on the road's width. This hidden unit would cooperate with hidden units that detect the right road edge to determine the correct travel direction in any particular situation.

## DISCUSSION AND EXTENSIONS

The distinct representations developed for different circumstances illustrate a key advantage provided by neural networks for autonomous navigation. Namely, in this paradigm the data, not the programmer, determines the salient image features crucial to accurate road navigation. From a practical standpoint, this data responsiveness has dramatically sped ALVINN's development. Once a realistic artificial road generator was developed, back-propagation produced in half an hour a relatively successful road following system. It took many months of algorithm development and parameter tuning by the vision and autonomous navigation groups at CMU to reach a similar level of performance using traditional image processing and pattern recognition techniques.

More speculatively, the flexibility of neural network representations provides the possibility of a very different type of autonomous navigation system in which the salient sensory features are determined for specific driving conditions. By interactively training the network on real road images taken as a human drives the NAVLAB, we hope to develop a system that adapts its processing to accommodate current circumstances. This is in contrast with other autonomous navigation systems at CMU [Thorpe et al., 1987] and elsewhere [Dunlay & Seida, 1988] [Dickmanns & Zapp, 1986] [Kuan et al., 1988]. Each of these implementations has relied on a fixed, highly structured and therefore relatively inflexible algorithm for finding and following the road, regardless of the conditions at hand.

There are difficulties involved with training "on-the-fly" with real images. If the network is not presented with sufficient variability in its training exemplars to cover the conditions it is likely to encounter when it takes over driving from the human operator, it will not develop a sufficiently robust representation and will perform poorly. In addition, the network must not solely be shown examples of accurate driving, but also how to recover (i.e. return to the road center) once a mistake has been made. Partial initial training on a variety of simulated road images should help eliminate these difficulties and facilitate better performance.

Another important advantage gained through the use of neural networks for autonomous navigation is the ease with which they assimilate data from independent sensors. The current ALVINN implementation processes data from two sources, the video camera and the laser range finder. During training, the network discovers how information from each source relates to the task, and weights each accordingly. As an example, range data is in some sense less important for the task of road following than is the video data. The range data contains information concerning the position of obstacles in the scene, but nothing explicit about the location of the road. As a result, the range data is given less significance in the representation, as is illustrated by the relatively small

magnitude weights from the range finder retina in the weight diagrams. Figures 4 and 5 illustrate that the range finder connections do correlate with the connections from the video camera, and do contribute to choosing the correct travel direction. Specifically, in both figures, obstacles located outside the area in which the hidden unit expects the road to be located increase the hidden unit's activation level while obstacles located within the expected road boundaries inhibit the hidden unit. However the contributions from the range finger connections aren't necessary for reasonable performance. When ALVINN was tested with normal video input but an obstacle-free range finder image as constant input, there was no noticeable degradation in driving performance. Obviously under off-road driving conditions obstacle avoidance would become much more important and hence one would expect the range finder retina to play a much more significant role in the network's representation. We are currently working on an off-road version of ALVINN to test this hypothesis.

Other current directions for this project include conducting more extensive tests of the network's performance under a variety of weather and lighting conditions. These will be crucial for making legitimate performance comparisons between ALVINN and other autonomous navigation techniques. We are also working to increase driving speed by implementing the network simulation on the on-board Warp computer.

Additional extensions involve exploring different network architectures for the road following task. These include 1) giving the network additional feedback information by using Elman's [Elman, 1988] technique of recirculating hidden activation levels, 2) adding a second hidden layer to facilitate better internal representations, and 3) adding local connectivity to give the network a priori knowledge of the two dimensional nature of the input.

In the area of planning, interesting extensions include stopping for, or planning a path around, obstacles. One area of planning that clearly needs work is dealing sensibly with road forks and intersections. Currently upon reaching a fork, the network may output two widely discrepant travel directions, one for each choice. The result is often an oscillation in the dictated travel direction and hence inaccurate road following. Beyond dealing with individual intersections, we would eventually like to integrate a map into the system to enable global point-to-point path planning.

## CONCLUSION

More extensive testing must be performed before definitive conclusions can be drawn concerning the performance of ALVINN versus other road followers. We are optimistic concerning the eventual contributions neural networks will make to the area of autonomous navigation. But perhaps just as interesting are the possibilities of contributions in the other direction. We hope that exploring autonomous navigation, and in particular some of the extensions outlined in this paper, will have a significant impact on the field of neural networks. We certainly believe it is important to begin researching and evaluating neural networks in real world situations, and we think autonomous navigation is an interesting application for such an approach.

## Acknowledgements

This work would not have been possible without the input and support provided by Dave Touretzky, Joseph Tebelskis, George Gusciora and the CMU Warp group, and particularly Charles Thorpe, Jill Crisman, Martial Hebert, David Simon, and rest of the CMU ALV group.

This research was supported by the Office of Naval Research under Contracts N00014-87-K-0385, N00014-87-K-0533 and N00014-86-K-0678, by National Science Foundation Grant EET-8716324, by the Defense Advanced Research Projects Agency (DOD) monitored by the Space and Naval Warfare Systems Command under Contract N00039-87-C-0251, and by the Strategic Computing Initiative of DARPA, through ARPA Order 5351, and monitored by the U.S. Army Engineer Topographic Laboratories under contract DACA76-85-C-0003 titled "Road Following".

## References

[Dickmanns & Zapp, 1986] Dickmanns, E.D., Zapp, A. (1986) A curvature-based scheme for improving road vehicle guidance by computer vision. *"Mobile Robots"*, *SPIE-Proc. Vol. 727*, Cambridge, MA.

[Elman, 1988] Elman, J.L, (1988) Finding structure in time. Technical report 8801. Center for Research in Language, University of California, San Diego.

[Dunlay & Seida, 1988] Dunlay, R.T., Seida, S. (1988) Parallel off-road perception processing on the ALV. *Proc. SPIE Mobile Robot Conference*, Cambridge MA.

[Jackel et al., 1988] Jackel, L.D., Graf, H.P., Hubbard, W., Denker, J.S., Henderson, D., Guyon, I. (1988) An application of neural net chips: Handwritten digit recognition. *Proceedings of IEEE International Conference on Neural Networks*, San Diego, CA.

[Jordan, 1988] Jordan, M.I. (1988) Supervised learning and systems with excess degrees of freedom. COINS Tech. Report 88-27, Computer and Information Science, University of Massachusetts, Amherst MA.

[Kuan et al., 1988] Kuan, D., Phipps, G. and Hsueh, A.-C. Autonomous Robotic Vehicle Road Following. *IEEE Trans. on Pattern Analysis and Machine Intelligence, Vol. 10*, Sept. 1988.

[Pawlicki et al., 1988] Pawlicki, T.F., Lee, D.S., Hull, J.J., Srihari, S.N. (1988) Neural network models and their application to handwritten digit recognition. *Proceedings of IEEE International Conference on Neural Networks*, San Diego, CA.

[Pomerleau et al., 1988] Pomerleau, D.A., Gusciora, G.L., Touretzky, D.S., and Kung, H.T. (1988) Neural network simulation at Warp speed: How we got 17 million connections per second. *Proceedings of IEEE International Conference on Neural Networks*, San Diego, CA.

[Thorpe et al., 1987] Thorpe, C., Herbert, M., Kanade, T., Shafer S. and the members of the Strategic Computing Vision Lab (1987) Vision and navigation for the Carnegie Mellon NAVLAB. *Annual Review of Computer Science Vol. II*, Ed. Joseph Traub, Annual Reviews Inc., Palo Alto, CA.

[Waibel et al., 1988] Waibel, A., Hanazawa, T., Hinton, G., Shikano, K., Lang, K. (1988) Phoneme recognition: Neural Networks vs. Hidden Markov Models. *Proceedings from Int. Conf. on Acoustics, Speech and Signal Processing*, New York, New York.